# Non-rigid point set registration: Coherent Point Drift

**Andriy Myronenko      Xubo Song      Miguel Á. Carreira-Perpiñán**
Department of Computer Science and Electrical Engineering
OGI School of Science and Engineering
Oregon Health and Science University
Beaverton, OR, USA, 97006
`{myron, xubosong, miguel}@csee.ogi.edu`

## Abstract

We introduce Coherent Point Drift (CPD), a novel probabilistic method for non-rigid registration of point sets. The registration is treated as a Maximum Likelihood (ML) estimation problem with motion coherence constraint over the velocity field such that one point set moves coherently to align with the second set. We formulate the motion coherence constraint and derive a solution of regularized ML estimation through the variational approach, which leads to an elegant kernel form. We also derive the EM algorithm for the penalized ML optimization with deterministic annealing. The CPD method simultaneously finds both the non-rigid transformation and the correspondence between two point sets without making any prior assumption of the transformation model except that of motion coherence. This method can estimate complex non-linear non-rigid transformations, and is shown to be accurate on 2D and 3D examples and robust in the presence of outliers and missing points.

## 1   Introduction

Registration of point sets is an important issue for many computer vision applications such as robot navigation, image guided surgery, motion tracking, and face recognition. In fact, it is the key component in tasks such as object alignment, stereo matching, point set correspondence, image segmentation and shape/pattern matching. The registration problem is to find meaningful correspondence between two point sets and to recover the underlying transformation that maps one point set to the second. The "points" in the point set are features, most often the locations of interest points extracted from an image. Other common geometrical features include line segments, implicit and parametric curves and surfaces. Any geometrical feature can be represented as a point set; in this sense, the point locations is the most general of all features.

Registration techniques can be rigid or non-rigid depending on the underlying transformation model. The key characteristic of a rigid transformation is that all distances are preserved. The simplest non-rigid transformation is affine, which also allows anisotropic scaling and skews. Effective algorithms exist for rigid and affine registration. However, the need for more general non-rigid registration occurs in many tasks, where complex non-linear transformation models are required. Non-linear non-rigid registration remains a challenge in computer vision.

Many algorithms exist for point sets registration. A direct way of associating points of two arbitrary patterns is proposed in [1]. The algorithm exploits properties of singular value decomposition and works well with translation, shearing and scaling deformations. However, for a non-rigid transformation, the method performs poorly. Another popular method for point sets registration is the Iterative Closest Point (ICP) algorithm [2], which iteratively assigns correspondence and finds the least squares transformation (usually rigid) relating these point sets. The algorithm then redetermines the closest point set and continues until it reaches the local minimum. Many variants of ICP

have been proposed that affect all phases of the algorithm from the selection and matching of points to the minimization strategy [3]. Nonetheless ICP requires that the initial pose of the two point sets be adequately close, which is not always possible, especially when transformation is non-rigid [3].

Several non-rigid registration methods are introduced [4, 5]. The Robust Point Matching (RPM) method [4] allows global to local search and soft assignment of correspondences between two point sets. In [5] it is further shown that the RPM algorithm is similar to Expectation Maximization (EM) algorithms for the mixture models, where one point set represents data points and the other represents centroids of mixture models. In both papers, the non-rigid transform is parameterized by Thin Plate Spline (TPS) [6], leading to the TPS-RPM algorithm [4]. According to regularization theory, the TPS parametrization is a solution of the interpolation problem in 2D that penalizes the second order derivatives of the transformation. In 3D the solution is not differentiable at point locations. In four or higher dimensions the generalization collapses completely [7]. The M-step in the EM algorithm in [5] is approximated for simplification. As a result, the approach is not truly probabilistic and does not lead, in general, to the true Maximum Likelihood solution.

A correlation-based approach to point set registration is proposed in [8]. Two data sets are represented as probability densities, estimated using kernel density estimation. The registration is considered as the alignment between the two distributions that minimizes a similarity function defined by $L_2$ norm. This approach is further extended in [9], where both densities are represented as Gaussian Mixture Models (GMM). Once again thin-plate spline is used to parameterize the smooth non-linear underlying transformation.

In this paper we introduce a probabilistic method for point set registration that we call the Coherent Point Drift (CPD) method. Similar to [5], given two point sets, we fit a GMM to the first point set, whose Gaussian centroids are initialized from the points in the second set. However, unlike [4, 5, 9] which assumes a thin-plate spline transformation, we do not make any explicit assumption of the transformation model. Instead, we consider the process of adapting the Gaussian centroids from their initial positions to their final positions as a temporal motion process, and impose a motion coherence constraint over the velocity field. Velocity coherence is a particular way of imposing smoothness on the underlying transformation. The concept of motion coherence was proposed in the Motion Coherence Theory [10]. The intuition is that points close to one another tend to move coherently. This motion coherence constraint penalizes derivatives of all orders of the underlying velocity field (thin-plate spline only penalizes the second order derivative). Examples of velocity fields with different levels of motion coherence for different point correspondence are illustrated in Fig. 1.

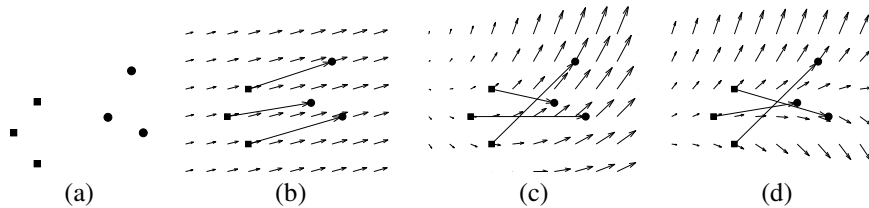

(a)           (b)           (c)           (d)

Figure 1: (a) Two given point sets. (b) A coherent velocity field. (c, d) Velocity fields that are less coherent for the given correspondences.

We derive a solution for the velocity field through a variational approach by maximizing the likelihood of GMM penalized by motion coherence. We show that the final transformation has an elegant kernel form. We also derive an EM algorithm for the penalized ML optimization with deterministic annealing. Once we have the final positions of the GMM centroids, the correspondence between the two point sets can be easily inferred through the posterior probability of the Gaussian mixture components given the first point set. Our method is a true probabilistic approach and is shown to be accurate and robust in the presence of outliers and missing points, and is effective for estimation of complex non-linear non-rigid transformations. The rest of the paper is organized as follows. In Section 2 we formulate the problem and derive the CPD algorithm. In Section 3 we present the results of CPD algorithm and compare its performance with that of RPM [4] and ICP [2]. In Section 4 we summarize the properties of CPD and discuss the results.

## 2 Method

Assume two point sets are given, where the template point set $\mathbf{Y} = (\mathbf{y}_1, \ldots, \mathbf{y}_M)^T$ (expressed as a $M \times D$ matrix) should be aligned with the reference point set $\mathbf{X} = (\mathbf{x}_1, \ldots, \mathbf{x}_N)^T$ (expressed as a $N \times D$ matrix) and $D$ is the dimension of the points. We consider the points in $\mathbf{Y}$ as the centroids of a Gaussian Mixture Model, and fit it to the data points $\mathbf{X}$ by maximizing the likelihood function. We denote $\mathbf{Y}_0$ as the initial centroid positions and define a continuous velocity function $v$ for the template point set such that the current position of centroids is defined as $\mathbf{Y} = v(\mathbf{Y}_0) + \mathbf{Y}_0$.

Consider a Gaussian-mixture density $p(\mathbf{x}) = \sum_{m=1}^{M} \frac{1}{M} p(\mathbf{x}|m)$ with $\mathbf{x}|m \sim \mathcal{N}(\mathbf{y}_m, \sigma^2 \mathbf{I}_D)$, where $\mathbf{Y}$ represents $D$-dimensional centroids of equally-weighted Gaussians with equal isotropic covariance matrices, and $\mathbf{X}$ set represents data points. In order to enforce a smooth motion constraint, we define the prior $p(\mathbf{Y}|\lambda) \propto \exp\left(-\frac{\lambda}{2}\phi(\mathbf{Y})\right)$, where $\lambda$ is a weighting constant and $\phi(\mathbf{Y})$ is a function that regularizes the motion to be smooth. Using Bayes theorem, we want to find the parameters $\mathbf{Y}$ by maximizing the posteriori probability, or equivalently by minimizing the following energy function:

$$E(\mathbf{Y}) = -\sum_{n=1}^{N} \log \sum_{m=1}^{M} e^{-\frac{1}{2}\left\| \frac{\mathbf{x}_n - \mathbf{y}_m}{\sigma} \right\|^2} + \frac{\lambda}{2}\phi(\mathbf{Y}) \tag{1}$$

We make the i.i.d. data assumption and ignore terms independent of $\mathbf{Y}$. Equation 1 has a similar form to that of Generalized Elastic Net (GEN) [11], which has shown good performance in non-rigid image registration [12]; note that there we directly penalized $\mathbf{Y}$, while here we penalize the transformation $v$. The $\phi$ function represents our prior knowledge about the motion, which should be smooth. Specifically, we want the velocity field $v$ generated by template point set displacement to be smooth. According to [13], smoothness is a measure of the "oscillatory" behavior of a function. Within the class of differentiable functions, one function is said to be smoother than another if it oscillates less; in other words, if it has less energy at high frequency. The high frequency content of a function can be measured by first high-pass filtering the function, and then measuring the resulting power. This can be represented as $\phi(v) = \int_{\mathbb{R}^d} |\tilde{v}(\mathbf{s})|^2 / \tilde{G}(\mathbf{s}) \, d\mathbf{s}$, where $\tilde{v}$ indicates the Fourier transform of the velocity and $\tilde{G}$ is some positive function that approaches zero as $\|\mathbf{s}\| \to \infty$. Here $\tilde{G}$ represents a symmetric low-pass filter, so that its Fourier transform $G$ is real and symmetric. Following this formulation, we rewrite the energy function as:

$$E(\tilde{v}) = -\sum_{n=1}^{N} \log \sum_{m=1}^{M} e^{-\frac{1}{2}\left\| \frac{\mathbf{x}_n - \mathbf{y}_m}{\sigma} \right\|^2} + \frac{\lambda}{2} \int_{\mathbb{R}^d} \frac{|\tilde{v}(\mathbf{s})|^2}{\tilde{G}(\mathbf{s})} d\mathbf{s} \tag{2}$$

It can be shown using a variational approach (see Appendix A for a sketch of the proof) that the function which minimizes the energy function in Eq. 2 has the form of the radial basis function:

$$v(\mathbf{z}) = \sum_{m=1}^{M} \mathbf{w}_m G(\mathbf{z} - \mathbf{y}_{0m}) \tag{3}$$

We choose a Gaussian kernel form for $G$ (note it is not related to the Gaussian form of the distribution chosen for the mixture model). There are several motivations for such a Gaussian choice: First, it satisfies the required properties (symmetric, positive definite, and $\tilde{G}$ approaches zero as $\|\mathbf{s}\| \to \infty$). Second, a Gaussian low pass filter has the property of having the Gaussian form in both frequency and time domain without oscillations. By choosing an appropriately sized Gaussian filter we have the flexibility to control the range of filtered frequencies and thus the amount of spatial smoothness. Third, the choice of the Gaussian makes our regularization term equivalent to the one in Motion Coherence Theory (MCT) [10]. The regularization term $\int_{\mathbb{R}^d} |\tilde{v}(\mathbf{s})|^2 / \tilde{G}(\mathbf{s}) \, d\mathbf{s}$, with a Gaussian function for $G$, is equivalent to the sum of weighted squares of all order derivatives of the velocity field $\int_{\mathbb{R}^d} \sum_{m=1}^{\infty} \frac{\beta^{2m}}{m! 2^m} (D^m v)^2$ [10, 13] , where $D$ is a derivative operator so that $D^{2m} v = \nabla^{2m} v$ and $D^{2m+1} v = \nabla(\nabla^{2m} v)$. The equivalence of the regularization term with that of the Motion Coherence Theory implies that we are imposing motion coherence among the points and thus we call our method the Coherent Point Drift (CPD) method. Detailed discussion of MCT can be found in [10]. Substituting the solution obtained in Eq. 3 back into Eq. 2, we obtain

Figure 2: Pseudo-code of CPD algorithm.

$$E(\mathbf{W}) = -\sum_{n=1}^{N} \log \sum_{m=1}^{M} e^{-\frac{1}{2}\left\| \frac{\mathbf{x}_n - \mathbf{y}_{0m} - \sum_{k=1}^{M} \mathbf{w}_k G(\mathbf{y}_{0k} - \mathbf{y}_{0m})}{\sigma} \right\|^2} + \frac{\lambda}{2} \operatorname{tr}\left(\mathbf{W}^T \mathbf{G} \mathbf{W}\right) \qquad (4)$$

where $\mathbf{G}_{M \times M}$ is a square symmetric Gram matrix with elements $g_{ij} = e^{-\frac{1}{2}\left\| \frac{\mathbf{y}_{0i} - \mathbf{y}_{0j}}{\beta} \right\|^2}$ and $\mathbf{W}_{M \times D} = (\mathbf{w}_1, \ldots, \mathbf{w}_M)^T$ is a matrix of the Gaussian kernel weights in Eq. 3.

**Optimization.** Following the EM algorithm derivation for clustering using Gaussian Mixture Model [14], we can find the upper bound of the function in Eq. 4 as (E-step):

$$Q(\mathbf{W}) = \sum_{n=1}^{N} \sum_{m=1}^{M} P^{\text{old}}(m|\mathbf{x}_n) \frac{\|\mathbf{x}_n - \mathbf{y}_{0m} - \mathbf{G}(m, \cdot)\mathbf{W}\|^2}{2\sigma^2} + \frac{\lambda}{2} \operatorname{tr}\left(\mathbf{W}^T \mathbf{G} \mathbf{W}\right) \qquad (5)$$

where $P^{\text{old}}$ denotes the posterior probabilities calculated using previous parameter values, and $\mathbf{G}(m, \cdot)$ denotes the $m^{th}$ row of $\mathbf{G}$. Minimizing the upper bound $Q$ will lead to a decrease in the value of the energy function $E$ in Eq. 4, unless it is already at local minimum. Taking the derivative of Eq. 5 with respect to $\mathbf{W}$, and rewriting the equation in matrix form, we obtain (M-step)

$$\frac{\partial Q}{\partial \mathbf{W}} = \frac{1}{\sigma^2} \mathbf{G}(\operatorname{diag}(\mathbf{P1}))(\mathbf{Y}_0 + \mathbf{G}\mathbf{W}) - \mathbf{PX}) + \lambda \mathbf{G}\mathbf{W} = 0 \qquad (6)$$

where $\mathbf{P}$ is a matrix of posterior probabilities with $p_{mn} = e^{-\frac{1}{2}\left\| \frac{\mathbf{y}_m^{\text{old}} - \mathbf{x}_n}{\sigma} \right\|^2} / \sum_{m=1}^{M} e^{-\frac{1}{2}\left\| \frac{\mathbf{y}_m^{\text{old}} - \mathbf{x}_n}{\sigma} \right\|^2}$. The $\operatorname{diag}(\cdot)$ notation indicates diagonal matrix and $\mathbf{1}$ is a column vector of all ones. Multiplying Eq. 6 by $\sigma^2 \mathbf{G}^{-1}$ (which exists for a Gaussian kernel) we obtain a linear system of equations:

$$(\operatorname{diag}(\mathbf{P1}))\mathbf{G} + \lambda \sigma^2 \mathbf{I})\mathbf{W} = \mathbf{PX} - \operatorname{diag}(\mathbf{P1})\mathbf{Y}_0 \qquad (7)$$

Solving the system for $\mathbf{W}$ is the M-step of EM algorithm. The E step requires computation of the posterior probability matrix $\mathbf{P}$. The EM algorithm is guaranteed to converge to a local optimum from almost any starting point. Eq. 7 can also be obtained directly by finding the derivative of Eq. 4 with respect to $\mathbf{W}$ and equating it to zero. This results in a system of nonlinear equations that can be iteratively solved using fixed point update, which is exactly the EM algorithm shown above. The computational complexity of each EM iteration is dominated by the linear system of Eq. 7, which takes $\mathcal{O}(M^3)$. If using a truncated Gaussian kernel and/or linear conjugate gradients, this can be reduced to $\mathcal{O}(M^2)$.

**Robustness to Noise.** The use of a probabilistic assignment of correspondences between point sets is innately more robust than the binary assignment used in ICP. However, the GMM requires that each data point be explained by the model. In order to account for outliers, we add an additional uniform pdf component to the mixture model. This new component changes posterior probability matrix $\mathbf{P}$ in Eq. 7, which now is defined as $p_{mn} = e^{-\frac{1}{2}\left\| \frac{\mathbf{y}_m^{\text{old}} - \mathbf{x}_n}{\sigma} \right\|^2} / (\frac{(2\pi\sigma^2)^{\frac{D}{2}}}{a} + \sum_{m=1}^{M} e^{-\frac{1}{2}\left\| \frac{\mathbf{y}_m^{\text{old}} - \mathbf{x}_n}{\sigma} \right\|^2})$, where $a$ defines the support for the uniform pdf. The use of the uniform distribution greatly improves the noise.

**Free Parameters.** There are three free parameters in the method: $\lambda, \beta$ and $\sigma$. Parameter $\lambda$ represents the trade off between data fitting and smoothness regularization. Parameter $\beta$ reflects the strength

of interaction between points. Small values of $\beta$ produce locally smooth transformation, while large values of $\beta$ correspond to nearly pure translation transformation. The value of $\sigma$ serves as a capture range for each Gaussian mixture component. Smaller $\sigma$ indicates smaller and more localized capture range for each Gaussian component in the mixture model. We use deterministic annealing for $\sigma$, starting with a large value and gradually reducing it according to $\sigma = \alpha\sigma$, where $\alpha$ is annealing rate (normally between [0.92 0.98]), so that the annealing process is slow enough for the algorithm to be robust. The gradual reducing of $\sigma$ leads to a coarse-to-fine match strategy. We summarize the CPD algorithm in Fig. 2.

## 3   Experimental Results

We show the performance of CPD on artificial data with non-rigid deformations. The algorithm is implemented in Matlab, and tested on a Pentium4 CPU 3GHz with 4GB RAM. The code is available at `www.csee.ogi.edu/~myron/matlab/cpd`. The initial value of $\lambda$ and $\beta$ are set to 1.0 in all experiments. The starting value of $\sigma$ is 3.0 and gradually annealed with $\alpha = 0.97$. The stopping condition for the iterative process is either when the current change in parameters drops below a threshold of $10^{-6}$ or the number of iterations reaches the maximum of 150.

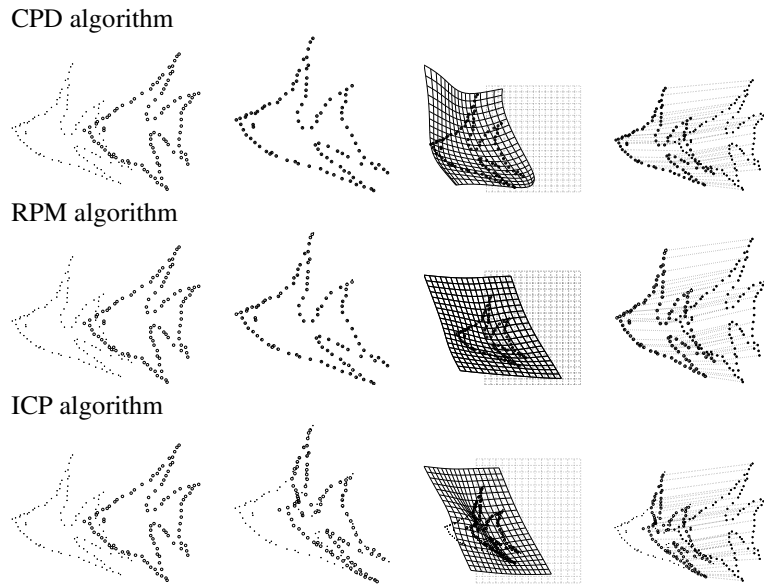

Figure 3: Registration results for the CPD, RPM and ICP algorithms from top to bottom. The first column shows template ($\circ$) and reference ($+$) point sets. The second column shows the registered position of the template set superimposed over the reference set. The third column represents the recovered underlying deformation . The last column shows the link between initial and final template point positions (only every second point's displacement is shown).

On average the algorithm converges in few seconds and requires around 80 iterations. All point sets are preprocessed to have zero mean and unit variance (which normalizes translation and scaling). We compare our method on non-rigid point registration with RPM and ICP. The RPM and ICP implementations and the 2D point sets used for comparison are taken from the TPS-RPM Matlab package [4].

For the first experiment (Fig. 3) we use two clean point sets. Both CPD and RPM algorithms produce accurate results for non-rigid registration. The ICP algorithm is unable to escape a local minimum. We show the velocity field through the deformation of a regular grid. The deformation field for RPM corresponds to parameterized TPS transformation, while that for CPD represents a motion coherent non-linear deformation. For the second experiment (Fig. 4) we make the registration problem more challenging. The fish head in the reference point set is removed, and random noise is added. In the template point set the tail is removed. The CPD algorithm shows robustness even in the area of

missing points and corrupted data. RPM incorrectly wraps points to the middle of the figure. We have also tried different values of smoothness parameters for RPM without much success, and we only show the best result. ICP also shows poor performance and is stuck in a local minimum.

For the 3D experiment (Fig. 5) we show the performance of CPD on 3D faces. The face surface is defined by the set of control points. We artificially deform the control point positions non-rigidly and use it as a template point set. The original control point positions are used as a reference point set. CPD is effective and accurate for this 3D non-rigid registration problem.

CPD algorithm

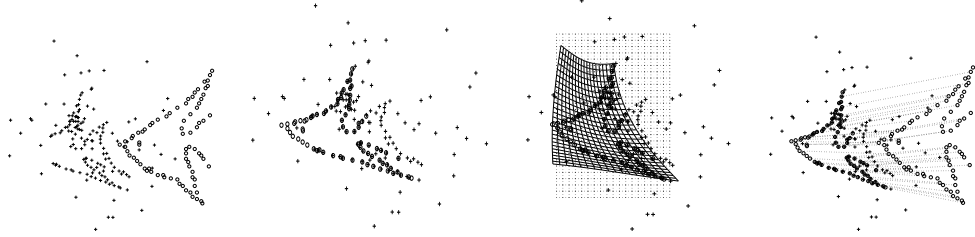

RPM algorithm

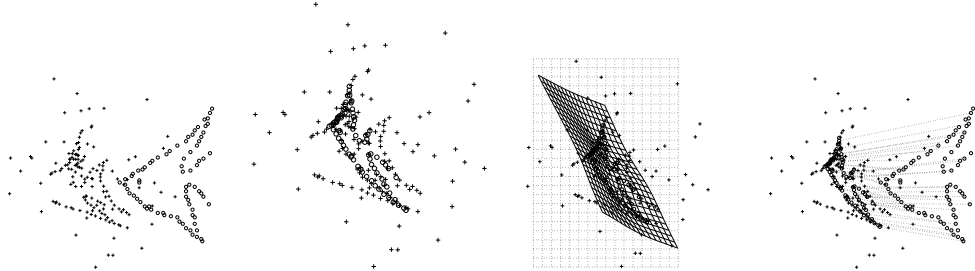

ICP algorithm

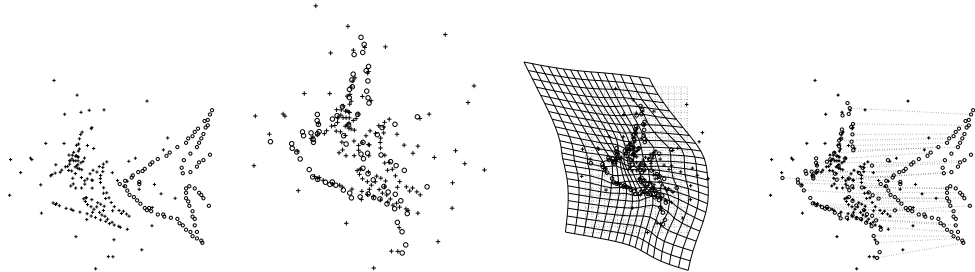

Figure 4: The reference point set is corrupted to make the registration task more challenging. Noise is added and the fish head is removed in the reference point set. The tail is also removed in the template point set. The first column shows template (○) and reference (+) point sets. The second column shows the registered position of the template set superimposed over the reference set. The third column represents the recovered underlying deformation. The last column shows the link between the initial and final template point positions.

## 4   Discussion and Conclusion

We intoduce Coherent Point Drift, a new probabilistic method for non-rigid registration of two point sets. The registration is considered as a Maximum Likelihood estimation problem, where one point set represents centroids of a GMM and the other represents the data. We regularize the velocity field over the points domain to enforce coherent motion and define the mathematical formulation of this constraint. We derive the solution for the penalized ML estimation through the variational approach, and show that the final transformation has an elegant kernel form. We also derive the EM optimization algorithm with deterministic annealing. The estimated velocity field represents the underlying non-rigid transformation. Once we have the final positions of the GMM centroids, the correspondence between the two point sets can be easily inferred through the posterior probability of

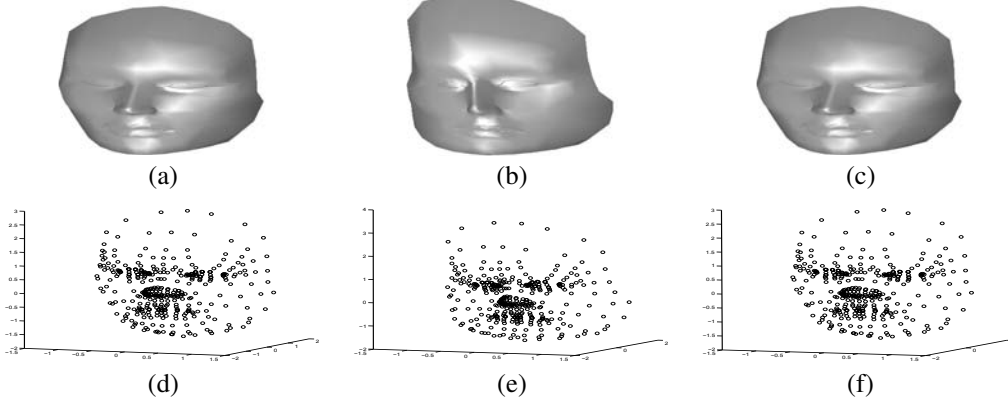

(a)                    (b)                    (c)

(d)                    (e)                    (f)

Figure 5: The results of CPD non-rigid registration on 3D point sets. (a, d) The reference face and its control point set. (b, e) The template face and its control point set. (c, f) Result obtained by registering the template point set onto the reference point set using CPD.

the GMM components given the data. The computational complexity of CPD is $\mathcal{O}(M^3)$, where $M$ is the number of points in template point set. It is worth mentioning that the components in the point vector are not limited to spatial coordinates. They can also represent the geometrical characteristic of an object (e.g., curvature, moments), or the features extracted from the intensity image (e.g., color, gradient). We compare the performance of the CPD algorithm on 2D and 3D data against ICP and RPM algorithms, and show how CPD outperforms both methods in the presence of noise and outliers. It should be noted that CPD does not work well for large in-plane rotation. Typically such transformation can be first compensated by other well known global registration techniques before CPD algorithm is carried out. The CPD method is most effective when estimating smooth non-rigid transformations.

## Appendix A

$$E = -\sum_{n=1}^{N} \log \sum_{m=1}^{M} e^{-\frac{1}{2}\left\|\frac{\mathbf{x}_n - \mathbf{y}_m}{\sigma}\right\|^2} + \frac{\lambda}{2}\int_{\mathbb{R}^d}\frac{|\tilde{v}(\mathbf{s})|^2}{\tilde{G}(\mathbf{s})}d\mathbf{s} \tag{8}$$

Consider the function in Eq. 8, where $\mathbf{y}_m = \mathbf{y}_{0m} + v(\mathbf{y}_{0m})$, and $\mathbf{y}_{0m}$ is the initial position of $\mathbf{y}_m$ point. $v$ is a continuous velocity function and $v(\mathbf{y}_{0m}) = \int_{\mathbb{R}^d} \tilde{v}(\mathbf{s})e^{2\pi i<\mathbf{y}_{0m},\mathbf{s}>}d\mathbf{s}$ in terms of its Fourier transform $\tilde{v}$. The following derivation follows [13]. Substituting $v$ into equation Eq. 8 we obtain:

$$E(\tilde{v}) = -\sum_{n=1}^{N} \log \sum_{m=1}^{M} e^{-\frac{1}{2}\left\|\frac{\mathbf{x}_n - \mathbf{y}_{0m} - \int_{\mathbb{R}^d} \tilde{v}(\mathbf{s})e^{2\pi i<\mathbf{y}_{0m},\mathbf{s}>}d\mathbf{s}}{\sigma}\right\|^2} + \frac{\lambda}{2}\int_{\mathbb{R}^d}\frac{|\tilde{v}(\mathbf{s})|^2}{\tilde{G}(\mathbf{s})}d\mathbf{s} \tag{9}$$

In order to find the minimum of this functional we take its functional derivatives with respect to $\tilde{v}$, so that $\frac{\delta E(\tilde{v})}{\delta \tilde{v}(\mathbf{t})} = 0, \forall \mathbf{t} \in \mathbb{R}^d$:

$$\frac{\delta E(\tilde{v})}{\delta \tilde{v}(\mathbf{t})} = -\sum_{n=1}^{N} \frac{\sum_{m=1}^{M} e^{-\frac{1}{2}\left\|\frac{\mathbf{x}_n - \mathbf{y}_m}{\sigma}\right\|^2}\frac{1}{\sigma^2}(\mathbf{x}_n - \mathbf{y}_m)\int_{\mathbb{R}^d}\frac{\delta \tilde{v}(\mathbf{s})}{\delta \tilde{v}(\mathbf{t})}e^{2\pi i<\mathbf{y}_{0m},\mathbf{s}>}d\mathbf{s}}{\sum_{m=1}^{M} e^{-\frac{1}{2}\left\|\frac{\mathbf{x}_n - \mathbf{y}_m}{\sigma}\right\|^2}} +$$

$$\frac{\lambda}{2}\int_{\mathbb{R}^d}\frac{\delta}{\delta \tilde{v}(\mathbf{t})}\frac{|\tilde{v}(\mathbf{s})|^2}{\tilde{G}(\mathbf{s})}d\mathbf{s} = -\sum_{n=1}^{N} \frac{\sum_{m=1}^{M} e^{-\frac{1}{2}\left\|\frac{\mathbf{x}_n - \mathbf{y}_m}{\sigma}\right\|^2}\frac{1}{\sigma^2}(\mathbf{x}_n - \mathbf{y}_m)e^{2\pi i<\mathbf{y}_0,\mathbf{t}>}}{\sum_{m=1}^{M} e^{-\frac{1}{2}\left\|\frac{\mathbf{x}_n - \mathbf{y}_m}{\sigma}\right\|^2}} + \lambda\frac{\tilde{v}(-\mathbf{t})}{\tilde{G}(\mathbf{t})} = 0$$

We now define the coefficients $\mathbf{a}_{mn} = \dfrac{e^{-\frac{1}{2}\left\|\frac{\mathbf{x}_n - \mathbf{y}_m}{\sigma}\right\|^2} \frac{1}{\sigma^2}(\mathbf{x}_n - \mathbf{y}_m)}{\sum_{m=1}^{M} e^{-\frac{1}{2}\left\|\frac{\mathbf{x}_n - \mathbf{y}_m}{\sigma}\right\|^2}}$, and rewrite the functional derivative as:

$$-\sum_{n=1}^{N}\sum_{m=1}^{M} \mathbf{a}_{mn} e^{2\pi i <\mathbf{y}_{0m},\mathbf{t}>} + \lambda \frac{\tilde{v}(-\mathbf{t})}{\tilde{G}(\mathbf{t})} = -\sum_{m=1}^{M}(\sum_{n=1}^{N} \mathbf{a}_{mn}) e^{2\pi i <\mathbf{y}_{0m},\mathbf{t}>} + \lambda \frac{\tilde{v}(-\mathbf{t})}{\tilde{G}(\mathbf{t})} = 0 \quad (10)$$

Denoting the new coefficients $\mathbf{w}_m = \frac{1}{\lambda}\sum_{n=1}^{N} \mathbf{a}_{mn}$, and changing $\mathbf{t}$ to $-\mathbf{t}$, we multiply by $\tilde{G}(\mathbf{t})$ on both sides of this equation, which results in:

$$\tilde{v}(\mathbf{t}) = \tilde{G}(-\mathbf{t})\sum_{m=1}^{M} \mathbf{w}_m e^{-2\pi i <\mathbf{y}_{0m},\mathbf{t}>} \qquad (11)$$

Assuming that $\tilde{G}$ is symmetric (so that its Fourier transform is real), and taking the inverse Fourier transform of the last equation, we obtain:

$$v(\mathbf{z}) = G(\mathbf{z}) * \sum_{m=1}^{M} \mathbf{w}_m \delta(\mathbf{z} - \mathbf{y}_{0m}) = \sum_{m=1}^{M} \mathbf{w}_m G(\mathbf{z} - \mathbf{y}_{0m}) \qquad (12)$$

Since $\mathbf{w}_m$ depend on $v$ through $a_{mn}$ and $\mathbf{y}_m$, the $\mathbf{w}_m$ that solve Eq. 12 must satisfy a self consistency equation equivalent to Eq. 7. A specific form of regularizer $\tilde{G}$ results in a specific basis function $G$.

## Acknowledgment

This work is partially supported by NIH grant NEI R01 EY013093, NSF grant IIS–0313350 (awarded to X. Song) and NSF CAREER award IIS–0546857 (awarded to Miguel Á. Carreira-Perpiñán).

## References

[1] G.L. Scott and H.C. Longuet-Higgins. An algorithm for associating the features of two images. *Royal Society London Proc.*, B-244:21–26, 1991.

[2] P.J. Besl and N. D. McKay. A method for registration of 3-d shapes. *IEEE Trans. Pattern Anal. Mach. Intell.*, 14(2):239–256, 1992.

[3] S. Rusinkiewicz and M. Levoy. Efficient variants of the ICP algorithm. *Third International Conference on 3D Digital Imaging and Modeling*, page 145, 2001.

[4] H Chui and A. Rangarajan. A new algorithm for non-rigid point matching. *CVPR*, 2:44–51, 2000.

[5] H. Chui and A. Rangarajan. A feature registration framework using mixture models. *IEEE Workshop on Mathematical Methods in Biomedical Image Analysis (MMBIA)*, pages 190–197, 2000.

[6] F. L. Bookstein. Principal warps: Thin-plate splines and the decomposition of deformations. *IEEE Trans. Pattern Anal. Mach. Intell.*, 11(6):567–585, 1989.

[7] R Sibson and G. Stone. Comp. of thin-plate splines. *SIAM J. Sci. Stat. Comput.*, 12(6):1304–1313, 1991.

[8] Y. Tsin and T. Kanade. A correlation-based approach to robust point set registration. *ECCV*, 3:558–569, 2004.

[9] B. Jian and B.C. Vemuri. A robust algorithm for point set registration using mixture of gaussians. *ICCV*, pages 1246–1251, 2005.

[10] A.L. Yuille and N.M. Grzywacz. The motion coherence theory. *Int. J. Computer Vision*, 3:344–353, 1988.

[11] M. Á. Carreira-Perpiñán, P. Dayan, and G. J. Goodhill. Differential priors for elastic nets. In *Proc. of the 6th Int. Conf. Intelligent Data Engineering and Automated Learning (IDEAL'05)*, pages 335–342, 2005.

[12] A. Myronenko, X Song, and M. Á. Carreira-Perpiñán. Non-parametric image registration using generalized elastic nets. *Int. Workshop on Math. Foundations of Comp. Anatomy: Geom. and Stat. Methods in Non-Linear Image Registration, MICCAI*, pages 156–163, 2006.

[13] F. Girosi, M. Jones, and T. Poggio. Regularization theory and neural networks architectures. *Neural Computation*, 7(2):219–269, 1995.

[14] C. M. Bishop. *Neural Networks for Pattern Recognition*. Oxford University Press, 1995.
